# Optimal Change-Detection and Spiking Neurons

**Angela J. Yu**
CSBMB, Princeton University
Princeton, NJ 08540
ajyu@princeton.edu

## Abstract

Survival in a non-stationary, potentially adversarial environment requires animals to detect sensory changes rapidly yet accurately, two oft competing desiderata. Neurons subserving such detections are faced with the corresponding challenge to discern "real" changes in inputs as quickly as possible, while ignoring noisy fluctuations. Mathematically, this is an example of a *change-detection* problem that is actively researched in the controlled stochastic processes community. In this paper, we utilize sophisticated tools developed in that community to formalize an instantiation of the problem faced by the nervous system, and characterize the Bayes-optimal decision policy under certain assumptions. We will derive from this optimal strategy an information accumulation and decision process that remarkably resembles the dynamics of a leaky integrate-and-fire neuron. This correspondence suggests that neurons are optimized for tracking input changes, and sheds new light on the *computational* import of intracellular properties such as resting membrane potential, voltage-dependent conductance, and post-spike reset voltage. We also explore the influence that factors such as timing, uncertainty, neuromodulation, and reward *should* and *do* have on neuronal dynamics and sensitivity, as the optimal decision strategy depends critically on these factors.

## 1 Introduction

Animals interacting with a changeable, potentially adversarial environment need to excel in the detection of changes in its sensory inputs. This detection, however, is riddled by the inherently competing goals of accuracy and speed. Due to the noisy and incomplete nature of sensory inputs, the animal can generally achieve more accurate detection by waiting for more sensory inputs. However, gathering this extra information incurs an opportunity cost, as the extra time can be used to gather more food, attract a mate, or escape a predator. Neurons subserving the detection process face a similar speed-accuracy trade-off. In this work, we aim to understand the computations performed by a neuron at the time-scale of single spikes. How sensitive a neuron is to each input spike should depend on the relative probabilities of the input representing noise and useful information, and the relative costs of mis-interpretation. We formulate the problem as an example of *change-detection*, and characterize the *optimal* decision policy in this context. The formal tools we utilize to formalize the change-detection problem are built upon work in the area of controlled stochastic processes. Controlled stochastic processes refer to decision-making in environments plagued not only by inferential uncertainty about the state of the world, but also uncertainty associated with the consequences of an action or decision on the world itself. Finding optimal decision policies for such processes is an actively researched problem in financial mathematics and operations research. As we will discuss below, neuronal change-detection is a prime example of such a problem.

In Sec. 2, we introduce the general framework of change-detection. In Sec. 3, we apply the framework to a specific scenario similar to that faced by the neuron, and characterize the optimal solution. In Sec. 3, we demonstrate that the optimal information accumulation and decision process has dynamics remarkably resembling that of a spiking neuron. We examine the computational import of certain intracellular properties, characterize the input-output firing rate relationship, and extend the framework of multi-source detection. In Sec. 4, we explore the behavioral consequences of opti-

mal change-detection and examine issues such as the speed-accuracy trade-off, temporal and spatial cueing, and neuromodulation.

## 2   A Bayesian Formulation of the Change-Detection Problem

**The Generative Model**
Suppose we have sequential inputs $x_1, x_2, \ldots$, which are generated *iid* by a distribution $f_0(x)$ before time $\theta \in \{0, 1, \ldots\}$, and by a distribution $f_1(x)$ afterwards, where the random variable (**r.v.**) $\theta$ denotes the sudden, hidden change time. $\theta$ has an initial probability $P(\theta = 0) = q_0$, and a geometric distribution thereafter: $P(\theta = t) = (1 - q_0)(1 - q)^{t-1}q$, $t > 0$. The change-detection problem is concerned with finding the optimal decision policy for reporting the change from $f_0$ to $f_1$ as early as possible while minimizing false-alarms [1]. A decision policy $\pi$ is a mapping, possibly stochastic, from all observations made so far to the control (or action) set, $\pi(\mathbf{x}_t \triangleq \{x_1, \ldots, x_t\}) \mapsto \{a_1, a_2\}$. The action $a_1$ terminates the observation process and reports $\theta \le t$, and $a_2$ continues the observation for another time step. Every unique decision policy is identified by a corresponding **r.v.** of stopping times $\tau \in \{0, 1, \ldots\}$. In the following, we will use $\pi$ and $\tau$ interchangeably to refer to a policy.

**The Loss Function**
Following convention [2], we assume a loss function linear in false alarms and detection delay:

$$l_\pi(\theta, \tau) = \mathbf{1}_{\{\tau < \theta; \pi\}} + \mathbf{1}_{\{\tau \ge \theta; \pi\}}c(\tau - \theta) \tag{1}$$

where $\mathbf{1}$ is the indicator function, and $c > 0$ is a constant that specifies the relative importance of speed and accuracy. The total loss is the expectation of this loss function over $\theta$ and $\tau$:

$$L_\pi \triangleq \langle l_\pi(\theta, \tau); \pi \rangle = \sum_{\theta=0}^{\tau=\infty} \left( \sum_{\tau=0}^{\theta-1} P(\theta, \tau) + \sum_{\tau=\theta}^{\infty} c(\tau - \theta)P(\theta, \tau) \right) = P(\tau < \theta) + c\langle(\tau - \theta)^+\rangle \tag{2}$$

An optimal policy $\pi^*$ minimizes $L_\pi$. Due to the linear loss in detection delay, the expected loss blows up for all policies that do not stop almost surely (**a.s.**; probability=1) in finite time; therefore, we restrict the optimization problem in the following to the class of almost-surely finite-time policies. Using the notation $P_t \triangleq P(\theta \le t | \mathbf{x}_t)$, we have the following:

$$P(\theta > \tau) = \sum_{t=0}^{\infty} P(\tau = t, \theta > \tau) = \sum_{t=0}^{\infty} \int P(\theta > \tau | \mathbf{x}_\tau)P(\tau = t | \mathbf{x}_t)p(\mathbf{x}_t)d\mathbf{x}_t = \langle 1 - P_\tau \rangle_{\tau, \mathbf{x}_\tau}$$

$$\langle(\tau - \theta)^+\rangle = \sum_{t=0}^{\infty} \langle \mathbf{1}_{\{\tau > t\}} \cdot \mathbf{1}_{\{\theta \le t\}} \rangle_{\theta, \tau} = \sum_{\tau=0}^{\infty} \sum_{t=0}^{\tau-1} P(\tau)P(\theta \le t) = \sum_{\tau=0}^{\infty} P(\tau) \sum_{t=0}^{t-1} \langle P_t \rangle_{\theta, \mathbf{x}_t} = \langle \sum_{t=0}^{\tau-1} P_t \rangle_{\theta, \mathbf{x}_t, \tau}$$

The cumulative posterior probability $P_\tau$ at the detection time $\tau$, therefore, is the critical factor in loss evaluation and policy optimization:

$$L_\pi = \langle c\Sigma_{k=0}^{\tau-1} P_k + (1 - P\tau) \rangle_{\theta, P_k, \tau; \pi} . \tag{3}$$

Bayes Rule gives us the iterative update rule for the cumulative posterior $P_t \triangleq P(\theta \le t | \mathbf{x}_t)$,

$$P_{t+1} = \frac{(P_t + (1 - P_t)q)f_1(x_{t+1})}{(P_t + (1 - P_t)q)f_1(x_{t+1}) + (1 - P_t)(1 - q)f_0(x_{t+1})} , \quad P_0 = q_0 . \tag{4}$$

$P_{t+1}$ is a deterministic function of $P_t$ and $x_{t+1}$, but appears to take a stochastic trajectory since $x_{t+1}$ is an i.i.d.-distributed **r.v.** The expectation of $\langle P_{t+1} | \mathbf{x}_t \rangle$ is $P_t + (1 - P_t)q$. We also define the monotonically related *posterior ratio* $\Phi_t = \frac{P_t}{1 - P_t}$, which has the update rule

$$\Phi_{t+1} = \frac{f_1(x_{t+1})(\Phi_t + q)}{f_0(x_{t+1})(1 - q)} , \quad \Phi_0 = \frac{q}{1 - q} . \tag{5}$$

**Optimal Policy: Threshold Crossing**
In order to optimize over the space of *all* possible stopping rules (policies), we define the following: (1) the *conditional termination cost*, $C_t$, associated with stopping at time $t$ after observing $\mathbf{x}_t$: $C_t \triangleq c\sum_{i=0}^{t-1} P_i + (1 - P_t)$; (2) the *minimal conditional cost*, $\gamma_t$, to be expected after observation $\mathbf{x}_t$: $\gamma_t \triangleq ess\ inf_\tau \langle C_\tau | \mathbf{x}_t \rangle$, where $\tau$ ranges over all stopping rules that terminate no earlier than $t$, and

the expectation is taken over all future observations (which can be a function of the decision taken at every time step); (3) *ess inf*, the largest (**a.s.**) **r.v.** less than (**a.s.**) every **r.v.** $X_n$, $n \in N$.

As an example of Bellman's Equation, $\gamma_t$ satisfies the dynamic programming equation $\gamma_t = \min\{C_t, \langle \gamma_{t+1} | \mathbf{x}_t \rangle\}$, and that the stationary, deterministic stopping rule $\tau^* = \min\{t \geq 1 | \gamma_t = C_t\}$ achieves optimality (Eq. 2). This implies that the optimal policy consists of a stopping region $S \subset [0, 1]$ and a continuation region $C = [0, 1] \setminus S$, such that $\pi(P_t : P_t \in S) = a_1$ and $\pi(P_t : P_t \in C) = a_2$. We will state and prove a useful theorem below, which will imply that $C$ and $S$ neatly fall into two contiguous blocks, such that the optimal policy requires the termination action as soon as $P_t$ exceeds some fixed threshold $B^*$ – *ie* the optimal policy is a *first-passage process* in $P_t$!

Before we present the theorem, we first introduce *the method of truncation*. The difficulty of solving the dynamic equation for $\gamma_t$ lies in its infinite recursiveness. If we can impose a *finite horizon* $T$ on $\tau$, then the finitely recursive relation $\gamma_t^T = \min\{C_t, \langle \gamma_{t+1}^T | \mathbf{x}_t \rangle\}$ has a corresponding finite-horizon optimal policy $\pi_T^*$, where $\gamma_T^T = C_T$. Taking the infinite limit $\gamma_t^\infty \triangleq \lim_{T \to \infty} \gamma_t^T$, it has been shown [2] that when the expected loss is finite (which is the case here, since the expression in Eq. 2 is finite for all decision policies that stop **a.s.** in finite time), $\gamma_t = \gamma_t^\infty$, and $\pi_T^*$ converges to the *infinite-horizon* optimal policy $\pi^*$. We also note the following self-evident lemma.

**Lemma.** Suppose $\{g_i(t)\}_{i \in I}$ is a family of decreasing functions in $t$, and $h(t) = \sum_i g_i(t) w_i(t)$, where $\sum_i w_i(t) = 1 \, \forall t$. If $g_i(t) \leq g_j(t)$ implies $w_i'(t) \geq w_j'(t)$, then $h(t)$ decreases with $t$.

**Theorem.** $C_t - \langle \gamma_{t+1}^T | \mathbf{x}_t \rangle$ is a decreasing function of $P_t$.
*Proof.* $C_{T-1} - \langle \gamma_T^T | \mathbf{x}_{T-1} \rangle$ decreases with $P_{T-1}$. Assume that the theorem holds for $t+1$, and note:

$$C_t - \langle \gamma_{t+1}^T | \mathbf{x}_t \rangle = -(c + q)P_t + q + \sum_i g_i w_i$$

where $g_i \triangleq \max(0, l_i)$, $l_i \triangleq C_{t+1} - \langle \gamma_{t+2}^T | \mathbf{x}_t, x_{t+1} = i \rangle$, and $w_i \triangleq P(x_{t+1} = i | \mathbf{x})$. $g_i$ decreases with $P_t$ for each $i$, since $l_i$ decreases with $P_{t+1}$ by the inductive hypothesis, and $P_{t+1}$ increases with $P_t$ by Eq. 4. Suppose $i, j$ are such that $f_1(i) - f_0(i) > f_1(j) - f_0(j)$, then $\Phi_{t+1}(i) > \Phi_{t+1}(j)$, and $P_{t+1}(i) > P_{t+1}(j)$, for any given $\mathbf{x}_t$. The inductive hypothesis implies $g_i \leq g_j$. Also note $dw_k/dP_t = (f_1(k) - f_0(k))(1 - q)$, so $dw_i/dP_t \geq dw_j/dP_t$. Thus, $C_t - \langle \gamma_{t+1}^T | \mathbf{x}_t \rangle$ decreases with $P_t$.

This theorem states that the cost of stopping at time $t$ relative to continuing gets smaller when it is more certain that $\theta \leq t$. This is true for any finite stopping time $T$ and therefore also for the infinite-horizon limit. If $C_t - \langle \gamma_{t+1} | \mathbf{x}_t \rangle$ is negative for some value of $P_t$, then the optimal policy is to select action $a_1$; this is also true for any larger values of $P_t$. Define $B^* \in [0, 1]$ as the lower bound of all such $P_t$, then the stopping and continuation regions have the form $[B^*, 1]$ and $[0, B^*)$, respectively.

Ideally, we would like to have an exact solution for the optimal policy as a function of the generative and cost parameters of the change-detection problem as defined in Sec. 1. While the explicit form of $B^*$ is not known, the theorem allows us to find the optimal policy numerically by evaluating and minimizing the empirical loss as a function of the decision threshold $B \in [0, 1]$.

## 3 Neuronal change-detection

In the following, we focus on the specific case where $f_0$ and $f_1$ are Bernoulli processes with respective rate parameters $\lambda_0$ and $\lambda_1$. This case resembles the problem faced by neurons, which receive sequential binary inputs (spike=1, no spike=0) with approximately Poisson statistics. The Bernoulli process is a discrete-time analog of the Poisson process, and obviates the problematic assumption (made by the Poisson model) that spikes could be fired infinitely close to one another. For now, we assume that the generative parameters $\lambda_1, \lambda_0, q_0, q$ and the cost parameter $c$ are known. We also assume, without loss of generality, that $\lambda_1 > \lambda_0$ (rate increases), since otherwise we can just redefine the inputs (0 or 1). When the parameters satisfy $c \geq (\lambda_1 - \lambda_0 - q(1 - \lambda_0))/(1 - \lambda_1)$, we have the explicit solution $B^* = q/(q = c)$, or $\Phi \geq q/c$ (proof omitted). This corresponds to the *one-step look-ahead* policy, and is optimal when the cost of detection is large or when the probability of the change taking place is very high. This turns out not to be a very interesting case as the detection process is driven to cross the threshold even in the absence of any input spikes.

Although we do not have an explicit solution for the optimal detection threshold $B^*$ in general, we can numerically compare different values of $B$ for any specific problem. Fig. 1(a) shows the empirical cost averaged over 1000 trials for different threshold values. For these particular parameters, the minimum is around $B = 0.65$, although the cost function is quite shallow for a large range of values of $B$ around the optimum, implying that performance is not particularly sensitive to relatively large perturbation around the optimal value.

**Repeated Change-Detection and Firing Rate**

From the problem formulation in sec. 2, it might seem like the framework only applies to detecting a single change, or multiple unrelated changes. However, the same policy formulation can apply to the case of repeated detection of changes, one after another, in a temporally contiguous fashion. As long as each detection event is generated from the same model parameters $(q, q_0, f_1, f_0)$, and the cost parameter $(c)$ remains constant, the threshold-crossing policy is still optimal in minimizing the empirical expected loss over these repeated events. The only generative parameter affected by the repetition is $q_0$, which represents the probability of the inputs already being generated from $f_1$ before the current observation process began. In this repeated detection scenario, $q_0$ should in general be high if the detection threshold $B^*$ is high, and low if $B^*$ is low. However, the strength of this coupling is tempered by (i) whether each detection termination resets the generative process, as happens when visual detection leads to saccades and thus the resetting of input statistics, and (ii) the amount of time elapsed during the refractory period after a detection spike. Fortunately, while $q_0$ is influenced by the detection policy, the optimization of the policy is not influenced by $q_0$, since it consists of comparing $C_t$ and $\langle \gamma_{t+1} | \mathbf{x}_t \rangle$ at every time step. This comparison does not depend on $q_0$, which simply adds a linear factor to both terms.

In this repeated firing scenario, where the number of spikes is relatively high relative to the frequency of changes, the loss function of Eq. 2 can be rewritten as $L_\pi = p_0 r_0 + c/r_1$, where $r_i$ is the mean firing rate when the inputs are generated from $f_i$, and $p_0$ is the fraction of time when $f_0$ is applicable (as opposed to $f_1$). In other words, if the rate of change is slow compared to neuronal firing rates, then optimal processing amounts to minimizing the "spontaneous" firing rate during $f_0$ and maximizing the "stimulus-evoked" firing rate during $f_1$.

**Optimality and Dynamics of Leaky Integrate-and-Fire**

Fig. 1(b) illustrates this concept of repeated firing. The top panel shows an example tracing of the dynamical variable $\Phi_t$ in the repeated optimal change-detection process. Whenever $\Phi_t$ reaches the threshold $0.65/(1-0.65)$ (or equivalently when $P_t$ reaches $0.65$, the optimal threshold as determined in the last section), a change is reported and the whole process resets to $\Phi_0$. The dynamics of $\Phi_t$ is remarkably similar to a leaky integrate-and-fire neuron. The bottom panel shows a raster plot of input and output spikes over 25 trials, and again the resemblance to spiking neurons is remarkable. Closer inspection indicates that the update rule for the posterior ratio in Eq. 5 indeed approximates the dynamics of a *leaky integrate-and-fire neuron* [3]. Let $a \triangleq \frac{f_1(x_t)}{(1-q)f_0(x_t)}$, we can rewrite Eq. 5 as

$$\Phi_t = a(\Phi_{t-1} + q) \tag{6}$$

When $x_t = 1$, $a = \frac{\lambda_1}{(1-q)\lambda_0} > 1$, $\Phi_t$ increases, and the rate of increase is larger when $\Phi_t$ itself is larger. This is reminiscent of the near-threshold dynamics of the Hodgkin-Huxley model, in which the *voltage-dependent* activation of sodium conductance drives the neuron to fire [4]. When $x_t = 0$, $\Phi_t$ converges to $\Phi_\infty^0 = f_1 q/(f_0(1-q) - f_1)$ (by Eq. 5), which is greater than 0 when $f_0(0)/f_1(0) \geq 1-q$. We can think of $\Phi_\infty^0$ as the *resting membrane potential*. Since $\Phi_\infty^0$ increases with $q$, it implies that the resting potential should be higher and closer to the firing threshold, making the neuron more sensitive to synaptic inputs, when there is a stronger expectation that a change is imminent.

**Relationship Between Input and Output Firing Rates**

We can also look at the input-output relationship at the firing-rate level. The state-dependent rate parameter $a$ has the expected values:

$$a_0 \triangleq \langle a | f_0 \rangle = \frac{1}{1-q} \qquad\qquad a_1 \triangleq \langle a | f_1 \rangle = \frac{1}{1-q} \frac{\lambda_1^2 + \lambda_0 - 2\lambda_0\lambda_1}{\lambda_0 - \lambda_0^2} .$$

Given Eqs. 5 and 6, we can write down an approximate, explicit expression for $\langle \Phi_t | f_i \rangle$:

$$\langle \Phi_t | f_i \rangle \approx a_i(\langle \Phi_{t-1} \rangle + q) = a_i^t \langle \Phi_0 \rangle + a_i q \sum_{k=0}^{t-1} a_i^k = a_i^t \langle \Phi_0 \rangle + \frac{a_i q (1 - a_i^t)}{1 - a_i} \approx a_i^t \left( \Phi_0 + \frac{q}{a_i - 1} \right) . \tag{7}$$

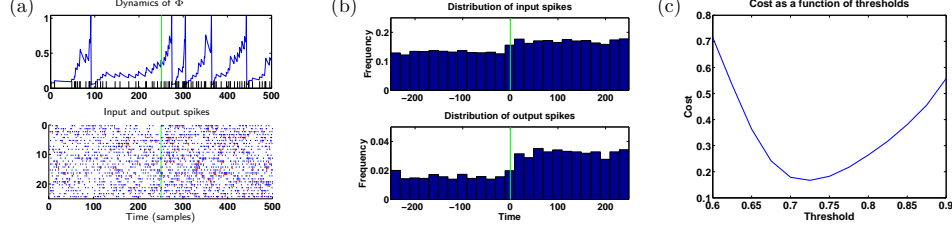

Figure 1: Optimal change-detection and dynamics. (a) The empirical average cost (over 1000 trials) has a single shallow minimum at $B = 0.65$. $\lambda_0 = 0.13$, $\lambda_1 = 0.17$, $q = 0.0125$, $q_0 = 0.05$, $c = 0.0005$; these parameters apply for the remainder of the paper unless otherwise specified. (b) Top panel: a typical example of the dynamics of $\Phi_t$ over time. Superimposed on $\Phi_t$ are the spikes, which are arbitrarily set to a fixed high value. Black bars near the bottom indicate input spikes. Green line indicates time of actual change. In this example, a chance flurry of input spikes near the start causes the optimal change-detector to fire; after the change, the increased input firing rate induces the change-detector fire much more frequently. Note that $\Phi_t$ decreases whenever there is a lull in input spikes. Bottom panel: Raster plot of input (blue) and output (red) spikes; both more frequent after the the change indicated by the green line. (c) Output spikes (bottom) increase frequency quickly after the increase in input spikes (top).

Given the decision threshold $B$, $\langle \Phi_{t_0} | f_0 \rangle = \langle \Phi_{t_1} | f_0 \rangle = B$, where $t_i$ is the average number of time steps it takes to reach the threshold for for $x_t = f_i$, and can be assumed to be $\gg 1$ (it takes many time steps of input integration to reach the threshold). We therefore have

$$a_0^{t_0}\left(\Phi_0 + \frac{q}{a_0-1}\right) = a_1^{t_1}\left(\Phi_1 + \frac{q}{a_1-1}\right) \implies a_1 = a_0^{t_0/t_1}\left(\frac{q/(a_0-1)+\Phi_0}{q/(a_1-1)+\Phi_0}\right)^{\frac{1}{t_1}} \approx a_0^{t_0/t_1}. \quad (8)$$

And therefore the ratio of the output firing rates, $r_i \triangleq 1/t_i$ for $i=1,2$, is

$$\frac{r_1}{r_0} = \frac{t_0}{t_1} = \frac{\log a_1}{\log a_0} = \frac{\log\frac{1}{1-q} + \log\frac{\lambda_1^2+\lambda_0-2\lambda_0\lambda 1}{\lambda_0-\lambda_0^2}}{\log\frac{1}{1-q}} = 1 + \frac{\log\frac{\lambda_1^2+\lambda_0-2\lambda_0\lambda 1}{\lambda_0-\lambda_0^2}}{\log\frac{1}{1-q}}. \quad (9)$$

Since the arguments of $\log$ in both the denominator and numerator are greater than 1, $r_1/r_0 > 1$. Therefore, when the input rates are such that $\lambda_1 > \lambda_0$, then the respective output rates are also such that $r_1 > r_0$. To see exactly how the output firing rate ratio changes as a function of the input rates, we define the function $g(\lambda_0, \lambda_1) \triangleq \frac{\lambda_1^2+\lambda_0-2\lambda_0\lambda 1}{\lambda_0-\lambda_0^2}$, and take its partial derivatives with respect to $\lambda_0$ and $\lambda_1$. Then we see that the *output firing ratio* Eq. 9 also increases with $\lambda_1$ and decreases with $\lambda_0$, consistent with intuitions. Fig. 1(c) shows the average detection/firing rate over time: the rise in output firing rate closely follows that in the input, despite the small change in the input firing rates.

**Multi-source change-detection**

So far, we have only considered the case of the Bernoulli inputs uniformly changing from one rate to another. However, sometimes the problem at hand is one of multi-source change-detection. For instance, a visual neuron detecting the onset of a stimulus might get inputs from up-stream neurons sensitive to stimuli with different properties (different colors, orientations, depth of view, etc.). Here, we extend our framework to the case of two independent sources of inputs, using an approach similar to that taken in [5]. The source $f^i, i \in \{1, 2\}$ emits observations $x_1^i, x_2^i, \ldots$ from a Bernoulli process that changes from rate $\lambda_0^i$ to $\lambda_1^i$ at an unknown time $\theta^i$, where $\theta^i$ is generated by a geometric distribution with parameter $q^i$, and the prior probability $P(\theta^i = 0)$ is $q_0^i$. The objective is to detect $\theta \triangleq \min(\theta_1, \theta_2)$ with the cost function specified as before (Eqs. 1-2).

Defining the individual posteriors $P_t^i \triangleq P(\theta^i \leq t | \mathbf{x}_t^i)$, where $\mathbf{x}_t^i \triangleq x_1^i, \ldots, x_t^i$, we have the following

$$P_t \triangleq P(\min(\theta^1, \theta^2) \leq t | \mathbf{x}_t^1, \mathbf{x}_t^2) = 1 - (1-P_t^1)(1-P_t^2) = P_t^1 + P_t^2 - P_t^1 P_t^2. \quad (10)$$

We can also define the corresponding overall posterior ratio

$$\Phi_t \triangleq P_t/(1-P_t) = \Phi_t^1 + \Phi_t^2 + \Phi_t^1 \Phi_t^2 \quad (11)$$

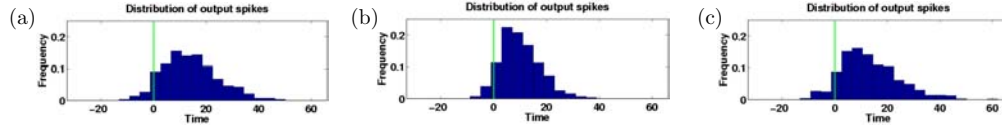

Figure 2: Effect of cueing on change-detection. (a) Distribution of first spikes for the optimal stopping policy; spikes aligned to time 0 when the actual change takes place. (b) This distribution is significantly tightened with mean brought closer to the actual change, when there is extra temporal information about an imminent change ($q = .02$). (c) The distribution of spikes is also slightly tightened and brought closer to the actual change time, when there is stronger prior probability of a stimulus appearing ($q_0 = .1$), as during special cueing. The effect is smaller because the higher prior leads to false alarms as well as reducing detection delay.

as a function of the individual posterior ratios $\Phi_t^i \triangleq P_t^i/(1 - P_t^i)$. Following reasoning very close to that of Sec. 2, we can show that if the generative and cost parameters are such that $\Phi_t$ is lower-bounded by $\Phi_\infty^0$ for $t \gg 1$, then the optimal stopping/detection policy is to terminate at the smallest $t$, such that $\Phi_t = \Phi_t^1 + \Phi_t^2 + \Phi_t^1 \Phi_t^2 \geq (q^1 + q^2 - q^1 q^2)/c$. Despite the generative independence of the two Bernoulli processes, we note that the optimal policy is *different* from the naïve strategy of running two single-source change-detectors, and report a change as soon as one of them reports a change. To see this, consider the case when $\Phi_t^1 = q^1/c$, but $\Phi_t^2 \approx 0$, so that $\Phi_t \approx \Phi_t^1 = q^1/c < (q^1 + q^2(1 - q^1))/c$. Therefore, the individual detector for process 1 would have reported a change, but the overall detector would not.

## 4    Optimal Change-Detection and Neuromodulation

A sizeable body of behavioral studies suggest that stimulus processing is influenced by cognitive factors such as knowledge about the timing of stimulus onset, or whether or not a stimulus would appear in a particular location. There is evidence that the neuromodulators norepinephrine [6], and acetylcholine [7] are respectively involved in those two aspects of stimulus processing. Separately, there is a rich literature on the effects of these various neuromodulators at the single-cell level [8]. Since we have here an explicit model of neuronal dynamics as a function of the statistical properties associated with the stimulus, we are ideally positioned to examine how these properties *should* affect the cellular properties, and whether the known behavioral consequences of neuromodulation are consistent with their observed effects at the cellular level.

If the system has some prior knowledge about the onset time of a stimulus, we can model the information accumulation process as starting shortly before the mean change-time, with a tight distribution over the random variable $\theta$. Making $q$ larger achieves both effects in our model. Fig. 2A shows the distribution of first spikes for 1000 trials; Fig. 2B shows that this distribution is more tightly clustered immediately after the actual change time $\theta$ for larger $q$. Experimentally, it has been observed that norepinephrine makes sensory neurons fire more vigorously to bottom-up sensory inputs [8]. It is also known from behavioral studies that a temporal cue improves detection performance, and that noradrenergic depletion diminishes this advantage [6].

If there is some prior knowledge about the stimulus being in a particular location, we can model this with a higher prior probability $q_0$ of the stimulus being present. This also has the effect of increasing the responsiveness of the change-detection process to input spikes (Fig. 2C), as well as making the detection (spiking) process more sensitive. It has been shown experimentally that a (correct) spatial cue improves stimulus detection, and that acetylcholine is implicated in this process [7], and that acetylcholine potentiates neurons and increases their responsiveness to sensory inputs [8].

## 5    Discussion

Responding accurately and rapidly to changes in the environment is a problem confronted by the brain at every level, from single neurons to behavior. In this work, we have presented a formal treatment of the change-detection problem and obtained important properties of the optimal policy – for a broad class of problems, the optimal detection algorithm is a threshold-crossing process based on the posterior probability of the change having taken place, which can be iteratively updated using Bayes'

Rule. Applying these ideas to the case of neurons that must rapidly and accurately detect changes in input spike statistics, we saw that the optimal algorithm yields dynamics remarkably similar to the intracellular dynamics of spiking neurons. This suggests that neurons are optimized for tracking discrete, abrupt changes in the inputs. The model yields insight into the computational import of cellular properties such as resting membrane potential, post-spike reset potential, voltage-dependent conductances, and the input-output spiking relationship. The basic framework was extended to examine the case of multi-source change-detection, a problem faced by a neuron tasked with detecting a stimulus when it could be one of two possible sub-categories. We also explored the computational consequences of spatial and temporal cueing on stimulus detection, and saw that the *behavioral* and *biophysical* effects of neuromodulation (*eg* by acetylcholine and norepinephrine) are consistent within the framework.

This novel framework for modeling single-neuron computations is attractive, as it suggests explicit *design principles* underlying neuronal dynamics, and not merely provides a descriptive model. Since the computational objects are well-specified at the outset, it provides a natural theoretica link between cellular properties and behavioral constraints. It is also appealing as a self-consistent and elegantly simple model of the computations taking place in single neurons. Every neuron in this scheme simply detects changes in its synaptic inputs, on a spike-to-spike time scale, and propagates its knowledge according to its own speed-accuracy trade-off. All that a down-stream neuron needs from this neuron for its own change-detection computations are this neuron's average firing rate in different states, the rate of change among these states, and the prior probability of of this neuron being in one of those states – all of these quantities can be learned over a longer time-scale. In particular, the down-stream neuron does not need to know about this neuron's inputs, its internal dynamics, its decision policy, its objective function, its model of the world, etc. In this scheme, more sophisticated computations can be achieved by pooling together the outputs of different neurons in various configurations – we explored this briefly with the example of multi-source change-detection. Another advantage of this framework is that it eliminates the boundary between *inference* and *decision*. In this scheme, neurons make inferences about their inputs and make decisions at *every* level of processing. It therefore obviates the problem of where in a hierarchical nervous system does the nature of the computation change from input-processing to decision-making.

While the incorporation of formal tools from controlled stochastic processes into the modeling of single-cell computations is a novel approach, this work is related to several other theoretical works. The idea of neurons processing and representing probabilistic information has received much attention in recent years, with most work focusing on the level of neuronal populations [9–12]. Theoretical work on the representation and processing of probabilistic information in single neurons are comparatively more rare. It has been suggested [13] that certain decision-making neurons may accumulate probabilistic information and spike when the evidence exceeds a certain threshold. However, it was typically assumed that the neurons already receive continuously-valued inputs that represent probabilistic information. Moreover, the tasks considered in these earlier works involved *stationary* discrimination, such that there was no explicit non-stationarity in the state of the world/inputs. We note that our framework is a *generalization* of the commonly studied 2AFC task, which is equivalent to setting the change probability $q$ to 0 in our model. Consistent with this characterization, our optimal policy is a generalization of the SPRT algorithm which is known to be optimal for stationary 2AFC discrimination [14].

One closely related piece of work proposed that single neurons track the *log* posterior ratio of the state of an underlying binary variable, and spike when the new inputs imply a value for this log posterior ratio that is sufficiently different from the neuron's current estimate based on previous inputs [15]. The key difference at the conceptual level is that this previous work focused on the explicit propagation of probabilistic information across neurons, thus introducing complications into processing and learning that are necessary to make this probabilistic knowledge consistent across neurons. Also, there was no explicit analysis of the optimality of the output spike generation process: how much of a discrepancy merits a spike, and how this depends on the relevant statistical and cost parameters. At the mechanistic level, having the membrane potential represent the *log* posterior ratio, as opposed to the posterior ratio, requires the dynamical update rule to involve exponentiation. While it was shown in that work that the dynamics is approximately leaky integrate-and-fire during steady state, it does not help the most interesting case, when the world is rapidly changing and the linear approximation is most detrimental. We showed in this work that there are good reasons for neurons *not* to integrate inputs linearly. The amount of new evidence provided by each input (spike

or not spike) at every time step is state-dependent, and *should* be so according to optimal information integration. This work suggests that the particular types of nonlinearity we see in neuronal dynamics are *desirable* from a computational point of view.

One important assumption we made in our model is that the cost of detection delay is linear in time, parameterized by the constant $c$. Without this assumption, the controlled dynamic process framework would not apply, as the decision policy would not only depend on a state variable, but on time in an explicit way. However, in general, there might not be a fixed $c$ that relates the trade-off between false alarms and detection delay. Intuitively, $c$ should be related to how much reward *could* be obtained per unit of time if the system were *not* engaged in prolonging the current observation process. In particular, if a new "trial" begins as soon as the current "trial" terminates, regardless of detection accuracy, then $c$ should be set to $P(\theta \leq \tau)/\langle \tau \rangle$, which also places the two cost terms in the same dimension. If we had analytical expressions for $P(\theta \leq \tau)$ and $\langle \tau \rangle$ as a function of the decision threshold $B$, then we could solve the optimization problem through the self-consistency constraint placed on the optimal threshold $B^*$ through its dependence on $c$. Unfortunately, there is no known analytical expressions for $P(\theta \leq \tau; B)$ and $\langle \tau; B \rangle$. Alternatively, one might still numerically obtain a value for a fixed detection threshold that incurs the lowest cost among all thresholds. There is no guarantee, however, that the optimal policy lives in this parameterized family of policies. It may be that the best fixed threshold policy is still far from optimal detection.

There are several important and exciting directions in which we plan to extend the current work. One is the consideration of more complex state transitions. In this work, we assumed that the state transition is always from $f_0$ to $f_1$. But in more general scenarios, the inputs are likely to revert back to $f_0$ before another transition into $f_1$, and so on. Thus, we need at least two populations of detectors, one that detects the onset ($f_0$ to $f_1$), and one that detects the offset ($f_1$ to $f_0$). Intuitively, there ought to be recurrent connections between them, to propagate and aggregate the total information about what states the inputs are in. A related problem is when the inputs can be in multiple ($> 2$) possible states, or even a continuous range of states, with complex transitions among these states. Another interesting question is what happens when we have a different or more complex distribution for the change variable $\theta$. We know, for instance, that animals are capable of utilizing independent temporal information about the mean and variance of the stimulus onset. In the geometric model we assumed, these two variables are coupled. Finally, we note that the formal framework we presented, that of optimal detection of changes in input statistics, is not only applicable to the level of single neuron, but also to systems and cognitive level problems. For example, certain problems in reinforcement learning, such as reversal learning and exploration versus exploitation in general, are also amenable to analysis by a similar approach. We intend to explore some of these problems in the future using similar formal tools from controlled dynamic processes.

### Acknowledgments

We thank Bill Bialek, Peter Dayan, Savas Dayanik, and Sophie Deneve for helpful discussions.

### References

[1] Shiryaev, A N (1978). *Optimal Stopping Rules*, Springer-Verlag, New York.

[2] Chow, Y S *et al* (1971). *Great Expectations: The Theory of Optimal Stopping*, Houghton Mifflin, Boston.

[3] Dayan, P & Abbott, L F (2001). *Theoretical Neuroscience*, MIT Press, Boston.

[4] Hodgkin, A L & Huxley, A F (1952). *J. Physiology* **117**: 500-44.

[5] Bayraktar, E & Poor, H V (2005). *44th IEEE Conf. on Decision and Control and Eur. Control Conference*.

[6] Witte, E A & Marrocco, R T (1997). Psychopharmacology **132**: 315-23.

[7] Phillips, J M, McAlonan, K, Robb, W G K & Brown, V (2000). Psychopharmacology **150**: 112-6.

[8] Gu, Q (2002). *Neuroscience* **111**: 815-35.

[9] Zemel, R S, Dayan, P, & Pouget, A (1998). *Neural Computation* **10**: 403-30.

[10] Sahani, M & Dayan, P (2003). *Neural Computation* **15**: 2255-79.

[11] Rao, R P (2004). *Neural Computation* **16**: 1-38.

[12] Yu, A J & Dayan, P (2005). *Advances in Neural Information Processing Systems* **17**.

[13] Gold, J I & Shadlen, M N (2002). *Neuron* **36**: 299-308.

[14] Wald, A & Wolfowitz, J (1948). *Ann. Math. Statisti.* **19**: 326-39.

[15] Deneve, S (2004). *Advances in Neural Information Processing Systems* **16**.
